# Animal-Bench: Benchmarking Multimodal Video Models for Animal-centric Video Understanding

Yinuo Jing[1], Ruxu Zhang[1], Kongming Liang[1,*], Yongxiang Li[2],
Zhongjiang He[2], Zhanyu Ma[1], Jun Guo[1]

[1]School of Artificial Intelligence, Beijing University of Posts and Telecommunications
[2]China Telecom Artificial Intelligence Technology Co. Ltd

{jingyinuo, zhangruxu, liangkongming, mazhanyu, guojun}@bupt.edu.cn
{liyx25, hezj}@chinatelecom.cn

## Abstract

With the emergence of large pre-trained multimodal video models, multiple benchmarks have been proposed to evaluate model capabilities. However, most of the benchmarks are human-centric, with evaluation data and tasks centered around human applications. Animals are an integral part of the natural world, and animal-centric video understanding is crucial for animal welfare and conservation efforts. Yet, existing benchmarks overlook evaluations focused on animals, limiting the application of the models. To address this limitation, our work established an animal-centric benchmark, namely Animal-Bench, to allow for a comprehensive evaluation of model capabilities in real-world contexts, overcoming agent-bias in previous benchmarks. Animal-Bench includes 13 tasks encompassing both common tasks shared with humans and special tasks relevant to animal conservation, spanning 7 major animal categories and 819 species, comprising a total of 41,839 data entries. To generate this benchmark, we defined a task system centered on animals and proposed an automated pipeline for animal-centric data processing. To further validate the robustness of models against real-world challenges, we utilized a video editing approach to simulate realistic scenarios like weather changes and shooting parameters due to animal movements. We evaluated 8 current multimodal video models on our benchmark and found considerable room for improvement. We hope our work provides insights for the community and opens up new avenues for research in multimodal video models. Our data and code will be released at
`https://github.com/PRIS-CV/Animal-Bench`.

## 1   Introduction

With the rapid advancement of artificial intelligence technology, multimodal video large models [1, 2, 3, 4, 5, 6, 7, 8] have bridged the gap between video and language modalities. Leveraging extensive knowledge and powerful comprehension capabilities, these models are being applied across various areas, ushering in a new era of intelligence. The emergence of a new era of intelligence is also accompanied by the development of new evaluation benchmarks. In contrast to conventional single-task benchmarks, the new benchmarks incorporate a variety of tasks and seek to evaluate the model's intelligence across multiple dimensions. Current evaluation studies [3, 9, 10, 11, 12, 13] primarily arise from human daily needs, with a focus on human-centric application tasks, aiming to assess

---

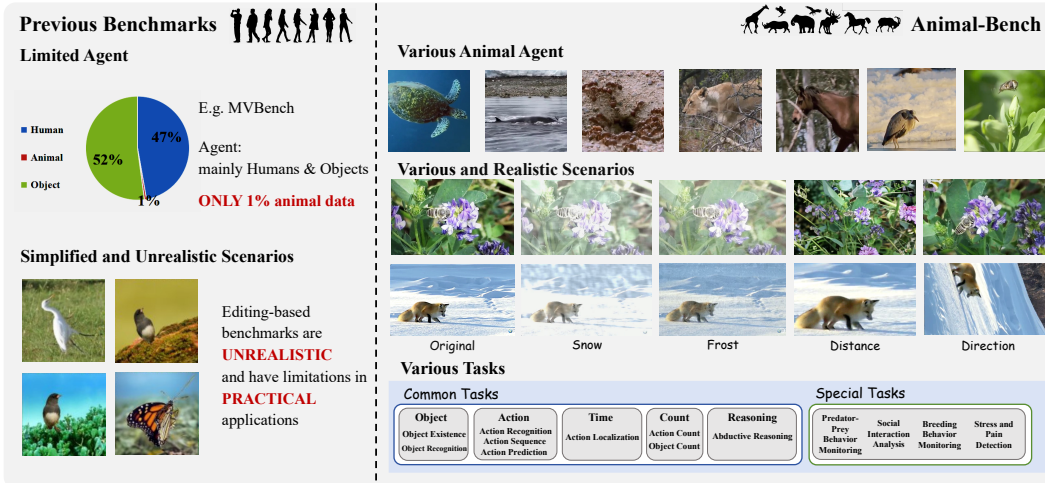

Figure 1: Previous benchmarks (left) relied on limited agent and the scenarios of editing-based benchmarks are unrealistic. Our proposed Animal-Bench (right) includes diverse animal agents, various realistic scenarios, and encompasses 13 different tasks.

model performance in common tasks encountered in human life. However, this approach confines their applicative scope to tasks centered around humans. In the broader real-world context, animals represent indispensable constituents of ecosystems [14]. They participate in vital processes such as pollination, seed dispersal, and nutrient cycling, which are essential for environmental conservation and the maintenance of biodiversity [15, 16, 17]. However, evaluations that focus on animal-centric tasks are entirely divergent from the current frameworks. As illustrated in Table 1, existing benchmarks that comprehensively evaluate model capabilities predominantly emphasize humans or objects, whereas animal-centric evaluation datasets can only assess the model's performance in few aspects. As shown in Figure 1, taking the popular benchmark MVBench [3] as an example, the main agents in the videos are humans and objects, with animal data accounting for only 1%. The inherent agent bias in these comprehensive benchmark datasets hampers our understanding of large models' ability to comprehend animal agents.

Research focusing on animal-centric evaluation overcomes the agent bias present in previous benchmarks, allowing us to assess model capabilities in a broader real-world framework, further exploring the potential applications of models and providing more valuable guidance for model optimization. The inherent diversity among animal species and the complexity of their habitats result in rich variability within animal videos, making animal-centric tasks highly challenging [18]. Evaluations on these demanding tasks can reveal the weaknesses of models in complex environments and analyze the robustness of multimodal video models against significant intra-class variations. Moreover, the applications of artificial intelligence in the field of animal studies are extensive [19, 20, 21, 22, 23, 24]. For instance, automated species counting [25, 26, 27] aids in tracking population dynamics in natural reserves, assessing overall ecosystem health, and significantly reducing human effort. Automated detection of animal stress and pain [28, 29, 30, 31, 32, 33] enables timely identification of potential issues, facilitating early treatment and ensuring animal welfare. Therefore, evaluations of models focused on animals are beneficial for further advancing the practical applications of artificial intelligence in the animal world. In summary, conducting animal-centric model evaluation plays a pivotal role in both model development and conservation efforts.

In this work, we propose Animal-Bench, a benchmark for evaluating multimodal video models in animal-centric video understanding. We choose tasks from two broad aspects: common tasks shared with human-centric benchmarks, covering aspects such as "object", "action", "time", "count", and "reasoning", and special tasks relevant to animal conservation. In total, we include 13 tasks spanning 7 major animal categories and 819 species, comprising a total of 41,839 data entries. To construct Animal-Bench, we devise a pipeline for automated data filtering and question-answer pair generation, reducing human effort and mitigating potential biases from human intervention. Furthermore, since our data primarily originates from the web, which typically features favorable

recording conditions, while real-world filming scenarios may involve harsh weather conditions such as snowy or frosty, or changes in shooting parameters, such as variations in camera distance and direction due to animal movements. To accommodate potential variations in model applications and simulate real-world filming conditions, we employ video editing methods for simulation. As illustrated in Figure 1, previous editing-based evaluation benchmark [34] generated images that did not correspond to real situations and were not applicable to our setting. We utilize a video editing approach based on the diffusion model [35] to simulate videos captured in realistic scenarios, thereby evaluating the robustness of multimodal video models. We evaluate 8 existing multimodal video models on our benchmark, identifying significant room for improvement. We hope our work can inspire advancements in multimodal video models development.

| Benchmarks | Dataset Properties | | | Tasks |
|---|---|---|---|---|
| | Label | QA Size | Agent(main) | |
| Video-MME [36] | Multi-Choice QA | 2700 QAs | Human & Object | object, action, attribute, position, count, time, reasoning, summarization, etc. |
| Video-Bench [11] | Multi-Choice QA | 15,033 QAs | Human & Object | action, object, attribute, position, count, time, reasoning, etc. |
| MVBench [3] | Multi-Choice QA | 4000 QAs | Human & Object | action, object, position, count, scene, pose, attribute, character and cognition |
| Animal Kingdom [37] | Classification | N/A | Animal | object, action, time |
| MammalNet [38] | Classification | N/A | Animal | object, action |
| Animal-Bench | Multi-Choice QA | 41,839 QAs | Animal | Common tasks: object, action, time, count, reasoning Special tasks: predator-prey behavior monitoring, social interaction analysis, breeding behavior monitoring and stress and pain detection |

Table 1: Comparison of existing video understanding benchmarks. In contrast to other benchmarks, Animal-Bench mitigates the limitations of prior video question-answering benchmarks that lack animal agents. The dataset is characterized by its richness and diversity, facilitating a comprehensive evaluation of models across multiple dimensions of performance.

## 2 Related Work

**MLLM Benchmarks**  Traditional evaluation benchmarks [39, 40, 41, 42, 43] typically only test the ability of models by a single task. With the rise of multi-modal large language models (MLLMs), new benchmarks [44, 13, 9] aimed to cover a wider range of evaluation aspects. For instance, Video-Bench [11] categorized the video language models' comprehension abilities into three levels: video-exclusive understanding, prior knowledge-based question-answering, and comprehension and decision-making. The Perception test [45] focused on skills (memory, abstraction, physics, semantics) and types of reasoning (descriptive, explanatory, predictive, counterfactual) across video, audio, and text modalities. These benchmarks all evaluated whether the models' capabilities were comprehensive enough by carefully choosing and dividing tasks, while overlooking the importance of data selection for evaluation. Other works considered how data selection influenced the evaluation process. For Image-LLM evaluation, MMBench [10] hierarchically subdivided the MLLM models' perception and cognition abilities and redesigned the QA pairs to better reflect capabilities of the models. For Video-LLM evaluation, MVBench [3] considered 20 challenging video tasks in its evaluations, selecting 200 test instances from open datasets for each task and redesigning the QAs. However, most existing benchmarks used human-centric data, predominantly featuring humans as the main agents and neglecting others. Our Animal-Bench believes that MLLMs should demonstrate good generalization abilities across different agents, thus designing a new benchmark and assessing the performance of MLLMs on animal-centric data.

**Editing-based Benchmarks**  In recent years, some works have employed editing techniques to process benchmarks, enhancing data diversity and evaluating model robustness. Regarding benchmarks based on image editing, LANCE [46] utilized image editing techniques to augment the test set with a suite of diverse, challenging, yet counterfactual examples for diagnosing the image recognition abilities of different models. D. Hendrycks's work [47] established a rigorous benchmark for testing image classifier robustness by introducing diverse types of corruptions and perturbations, including

noise, blur, weather effects, and digital distortions. For benchmarks based on video editing, Grover et al.[48] established an occluded dataset and further developed the benchmark to explore the impact of occlusion on action recognition models. Schiappa et al.'s work[49] proposed a robustness analysis by introducing 90 perturbations that reflected different real-world distribution shifts in their benchmark, offering insights into robust video action recognition. However, most of these works were counterfactual or only considered the effects of camera disturbances from the camera's viewpoint, neglecting factors such as scene characteristics (e.g., shooting distance and direction) that are likely to be encountered in real filming scenarios. Inspired by the aforementioned studies, Animal-Bench employs video editing techniques to create new animal videos under varying weather conditions and shooting parameters, presenting new demands and challenges for model robustness.

# 3 Methods

In this section, we will introduce the details of our Animal-Bench. In the first part, we describe how our Animal-Bench is designed in terms of task definition and the automated pipeline of data processing. The second part details our approach to editing videos in our benchmark, aiming to simulate realistic scenarios.

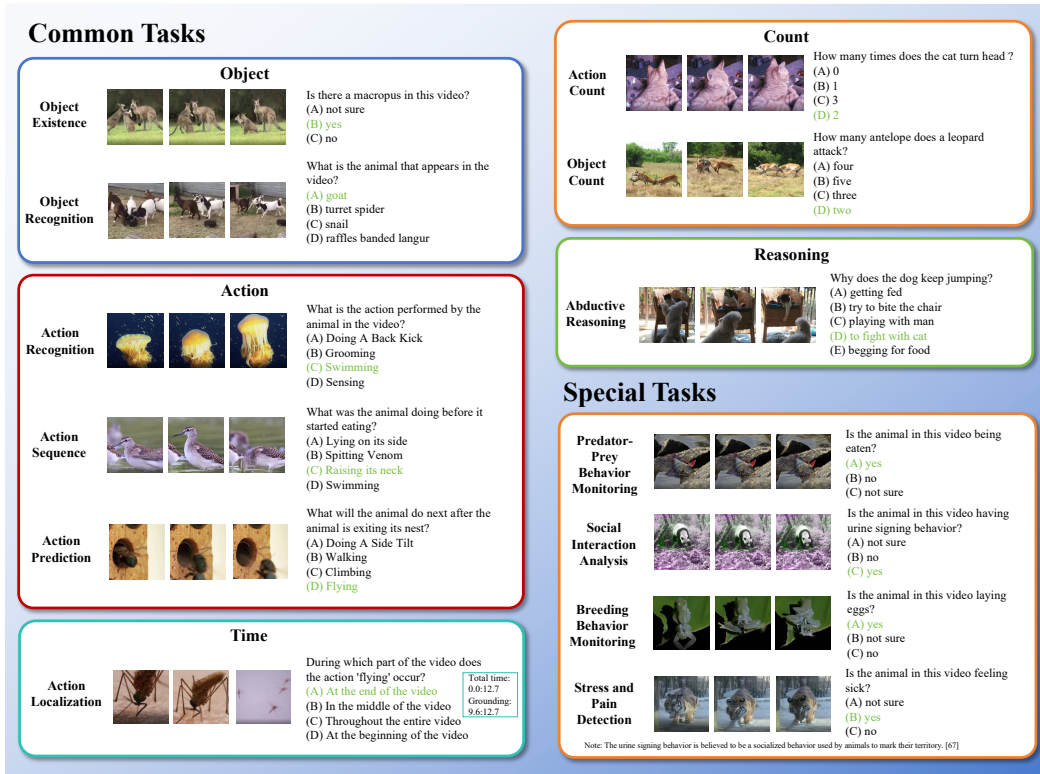

Figure 2: Example demonstrations of each task in Animal-Bench

## 3.1 Animal Bench: Animal-Centric Evaluation

### 3.1.1 Animal-Centric Tasks System

To evaluate the perceptual and cognitive abilities of multimodal large models on data where animals serve as the main agents, our Animal-Bench redefines tasks that were previously overlooked in human-centric benchmarks. First, we consider the common tasks from human-centric evaluations to assess the models' abilities in "object", "action", "time", "count", and "reasoning" on videos featuring animals. Additionally, from the perspective of application value, we identify specific tasks related to animals that are of greater interest to zoologists. For instance, we focus on models' ability to detect predator-prey behaviors [50], social interactions [51], breeding behaviors [52], and stress and

pain [53], thereby promoting better research and protection of animals. As shown in Figure 2, we designed the following tasks:

**Common Task**    Perception: Object. (1) Object Existence(OE): Judge whether a certain item exists during a particular video; (2) Object Recognition(OR): Determine the specific class of the object that appears in the video. Action. (3) Action Recognition(AR): Recognize the action performed by the animal based on a piece of video; (4) Action Sequence(AS): Infer the action of an animal before or after a certain action in chronological order; (5) Action Prediction(AP): Given a specific action and its starting and finishing time in the video, predict the subsequent action performed by the animal; (6) Action Localization(AL): Assess the start and end time of a specific action performed by the animals in the video. Count. (7) Action Count(AC): Calculate how many times an action has been performed in the video; (8) Object Count(OC): Calculate how many times an object appears in the video. Cognition: (9) Reasoning(RS): Logically infer why an event or a certain scenario occurred in the video.

**Specific Task**    (10) Predator-Prey Behavior Monitoring(PM): Detecting the interactions between predators and their prey that influence survival strategies, such as hunting techniques and evasion tactics; (11) Social Interaction Analysis(SA): Analyzing behaviors occurring between animals that affect social structure, communication, and cooperation; (12) Breeding Behavior Monitoring(BM): Monitoring activities related to reproduction, including mating and caregiving for offspring; (13) Stress and Pain Detection(PD): Detecting physiological and psychological responses to harmful stimuli or adverse conditions that impact the welfare and behavior of animals.

### 3.1.2   Animal-Centric Data Processing Pipeline

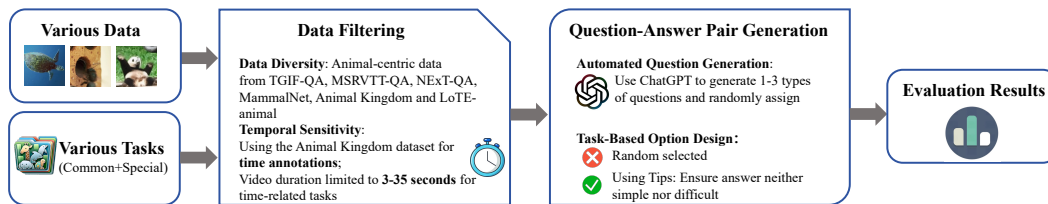

Figure 3: The diagram of our animal-centric data processing pipeline: Firstly, choose dataset and identify tasks, then establish rules to filter data, and finally automatically generate QA pairs.

We develop an automated pipeline for processing animal-centric data. Initially, we conduct data filtering on the existing dataset according to predefined task definitions. Subsequently, we formulate rules to automatically generate questions and options for all data, culminating in a dataset structured in the multiple-choice question answering (QA) format. The rationale for adopting the multiple-choice QA format [54, 55, 56] is twofold. Firstly, regarding difficulty, this format alleviates the challenge of delineating the scope of options inherent in open-ended QA formats, while still presenting a substantial challenge to capabilities of models. Secondly, from the point of evaluation fairness, the multiple-choice QA format facilitates the calculation of final evaluation accuracy and enables effective comparison of different models' performance.

**Data Filtering**    We first select data that aligns with the defined evaluation tasks. The principles for data selection are as follows. (1) Data diversity: Diverse data enables our evaluation to encompass a variety of complex situations, avoiding biases caused by single species or single dataset. (2) Temporal sensitivity: The temporal sensitivity present in the data allows us to assess the model's temporal modeling capabilities, as appropriate video lengths facilitate accurate decision-making for multi-modal video models.

To ensure the diversity of data, our Animal-Bench dataset is sourced from 6 datasets, encompassing 7 major animal categories, including mammals, insects, reptiles, amphibians, birds, fishes, and sea animals, totaling 819 different animal species. Specifically, for tasks related to "action", "object" and "time", we primarily get the annotated data for animals, actions, and grounding from MammalNet [38] and Animal Kingdom [37]. For the "count" and "reasoning" tasks, our evaluation data is sourced from open datasets such as TGIF-QA [57], MSRVTT-QA [42], and NExT-QA [40], from which we

extract a substantial amount of data featuring animals as agents. For special tasks, due to the lack of annotated data, we select data with annotations that match the requirements of the tasks from Animal Kingdom, LoTE-animal [58], and MammalNet [38].

To ensure the temporal sensitivity of the data, we first select the Animal Kingdom dataset, which contains annotations relevant to video-grounding tasks. Specifically, it marks the time intervals of actions, facilitating our evaluation of tasks such as action prediction, action sequence, and action localization. Additionally, for tasks other than "object," we believe that correct answers cannot be obtained solely from spatial information but require temporal information. For these tasks, we constrain the video duration to between 3 to 35 seconds. This ensures that answering questions requires relevant temporal information without causing confusion for the model due to excessively long video lengths. Ultimately, the average duration of our benchmark videos is 14.61 seconds.

**QA Pair Generation**    We have meticulously designed the generation rules for both the questions and the options. Here is the detailed process of our QA pair generation.

Automated Question Generation: For each task's description, we use ChatGPT [59] to generate 3 types of questions and randomly assign one of the generated questions to each piece of data.

Task-Based Option Design: (1) Directly adopt from existing QA annotation: For count and reasoning tasks, the options are directly chosen from the annotated multi-choice QA dataset. (2) Automatic option design: For other data without QA annotations, we automatically convert the original annotations into multi-choice QA format. For object existence tasks and special tasks, our options were set as "yes", "no", and "not sure". While for other tasks, options besides the right answer should be neither simple nor difficult in order to reflect the real perceptual ability of evaluated MLLMs, thereby our options should not be randomly chosen from the dataset. Specifically, for action-related tasks, considering the long-tailed distribution in the dataset, our four options consist of the correct answer, two options from the top 50% of most frequent answers, and one option from the least frequent 50% answers. For object recognition task, besides the right answer, two options are sampled from different major animal categories and one option is sampled from the same major animal category. This design ensures a balance in options. Once the options are set, they are randomly shuffled to ensure robustness of our evaluation. For a discussion on the option design rules, please refer to Appendix B.

## 3.2   Realistic simulation based on video editing

First, we select different aspects to simulate, as follows:

- **Weather conditions.** Weather changes are common during outdoor filming. In this work, we choose snowy and frosty weather as the simulated conditions. Snow is a form of precipitation that visually obstructs. Frost forms when ice crystals coat lenses or windows.
- **Shooting parameters.** Due to the camera's placement in the natural habitat of animals, the shooting parameters may vary due to animal movements. In this work, we select shooting distance (affected by the movements of animals resulting in changes in proximity to the camera) and shooting direction (affected by animal collisions resulting in camera tilt) as the simulated shooting parameters.

**Simulating variations in outdoor weather conditions**    We take simulating snowy weather as an example. First, we assume that the snow layer $S$ is an image with random noise, which follows a normal distribution $S \sim \mathcal{N}(\mu, \sigma^2)$. Next, we resize the image to make the density and distribution of snow more uniform. Since snowflakes do not accumulate in dark areas but only in well-lit areas when they fall on an object's surface, we set a threshold $t$ and remove parts below it. Specifically, we can express it as:

$$S' = \begin{cases} \text{zoom}(S, f) & \text{if } \text{zoom}(S, f) \geq t. \\ 0 & \text{otherwise} \end{cases} \tag{1}$$

Finally, we apply a blur effect to the snow layer to soften its edges. The blur operation can be expressed as:

$$S_{blurred}(x, y) = \frac{1}{2\pi\sigma^2} \sum_{i=-r}^{r} \sum_{j=-r}^{r} e^{-\frac{i^2+j^2}{2\sigma^2}} \cdot S'(x - i, y - j), \tag{2}$$

where, $S_{\text{blurred}}(x, y)$ denotes the pixel value after blurring. $r$ denotes the blur radius. $\sigma$ represents the standard deviation of the Gaussian kernel. $i$ and $j$ represent the indices of the convolution kernel.

**Simulating variations in shooting parameters** We aim to simulate different shooting distances from the camera to animals, as well as shooting directions. To simulate proximity, we achieve this through central cropping of each frame. However, when simulating remoteness or different shooting directions, we face the challenge of lacking broader context outside the current frame. To address this issue and enhance the realism of our video evaluations, we leverage the capabilities of the Diffusion model [60] to perform outpainting for regions beyond the original scene. Furthermore, we have developed an automated video editing pipeline to streamline this process.

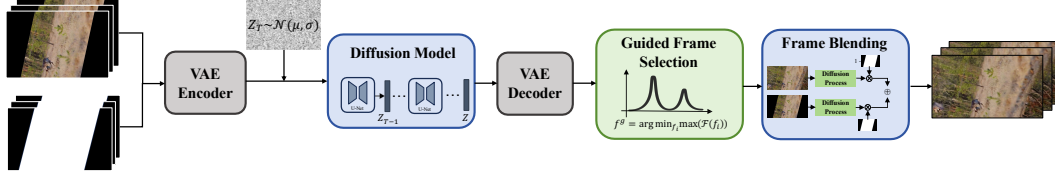

Figure 4: The diagram of simulation process for shooting parameters. Firstly, the transformed images along with their masks are encoded, and then passed through the diffusion model for denoising. After decoding, the final simulated video is obtained through the guided frame selection module and frame blending module.

For a video frame $f \in \mathcal{R}^{3 \times H \times W}$, we perform scaling or rotational transformations on it using an orthogonal matrix. Scaling and rotation matrices can be represented as follows,

$$T_S = \begin{pmatrix} \text{scale} & 0 \\ 0 & \text{scale} \end{pmatrix} \quad T_R = \begin{pmatrix} \cos\theta & -\sin\theta \\ \sin\theta & \cos\theta \end{pmatrix}. \tag{3}$$

Then we create a new blank canvas $f' \in \mathcal{R}^{3 \times H \times W}$, which has the same center position as the original image. Then the part of the original video frame that remains on the new canvas after transformations is:

$$f'(x', y') = \begin{cases} f\left(T^{-1}(x', y')^T\right), & \text{if } T^{-1}(x', y')^T \text{ is within the bounds of } f \\ 0, & \text{otherwise.} \end{cases} \tag{4}$$

We encode $f'$ and the mask of $f'$ using a variational autoencoder [61] to derive their respective image embeddings $y$ within the latent space. Subsequently, these embeddings, accompanied by initially sampled noise $\mathbf{z}_T \sim \mathcal{N}(0, \mathbf{I})$ drawn from a standard normal distribution, are jointly fed into denoising process. The reverse diffusion process is formulated as:

$$\mathbf{z}_{t-1} = \frac{1}{\sqrt{\alpha_t}}\left(\mathbf{z}_t - \frac{\beta_t}{\sqrt{1 - \bar{\alpha}_t}}\epsilon_\theta(\mathbf{z}_t, t, y)\right) + \sigma_t\mathbf{z}, \quad \mathbf{z} \sim \mathcal{N}(0, \mathbf{I}), \tag{5}$$

where $\alpha_t$ and $\beta_t$ are predefined diffusion coefficients. $\epsilon_\theta(\mathbf{z}_t, t, y)$ is the predicted noise. $\sigma_t$ is the noise standard deviation at each step. After obtaining the final latent space representation $\mathbf{z}_0$, the image is generated through the decoder, $\mathbf{x}_0 = \mathcal{D}(\mathbf{z}_0)$.

Due to the varying quality of outpainting generated by diffusion for each video frame, with some frames showing significantly better results than others, we observe that frames with smoother pixel transitions at the edges of the original image tend to have better outpainting quality. We hypothesize that the highest values in the image spectrum originate from the edges of the original outpainted image. Therefore, we select the frame with the smallest highest spectrum value as the guide image.

$$f^{\text{g}} = \arg\min_{f_i} \max(\mathcal{F}(f_i)), \quad \text{for} \quad f_i \in \{f_1, f_2, \ldots, f_n\} \tag{6}$$

If directly using the outpainting part in the $f^{\text{g}}$ would cause slight misalignment at the edges of other frames. Inspired by [34], we blend images at different noise levels along the diffusion process using diffusion models. Starting from the outpainting part $f^{\text{g}}$, at each stage, we perform a guided diffusion step with a latent variable $f_t^{\text{g}}$ to obtain $f_{t-1}^{\text{g}}$, while simultaneously obtaining a noisy version $f_{t-1}$ of the original frame $f$. The $f_{t-1}$ generated at this stage is a blend of two latent variables using a mask $M$, represented as follows.

$$f_{t-1} = M * f_{t-1} + (1 - M) * f_{t-1}^{\text{g}} \tag{7}$$

# 4 Experiments

In this section, we will sequentially introduce the details of our experimental implementation, the effectiveness of existing multimodal video models on our proposed evaluation benchmark, and their robustness on our editing-based realistic scenario simulation data. Finally, we will discuss the experimental results, hoping to provide guidance for model optimization.

## 4.1 Implementation details

We conducted all tests for multimodal video models on an NVIDIA RTX 4090 with 24GB of VRAM. To ensure fair comparisons, we standardize the 7B LLM backend versions used across all multi-modal video models tested during inference, thereby minimizing discrepancies in language proficiency due to differences in model sizes. Following the methodology outlined in [3], we establish a uniform system prompt and adopt the prompt-based model output matching strategy. All generated outputs successfully match the corresponding options. For each video, we sample 16 frames and resize them to (224, 224). During video editing, we utilize StableDiffusion-inpainting to expand the scenes beyond the captured footage and subsequently apply StableDiffusion-v1.5 for noise addition in frame blending.

## 4.2 Evaluation results

| | Multimodal Video Model | | | | | | | | | |
|---|---|---|---|---|---|---|---|---|---|---|
| Task | Random 95% confidence interval | mPLUG -Owl [44] | Video Chat [62] | Video -ChatGPT [63] | Video -LLaMA [8] | Valley [64] | Chat -UniVi [65] | Video -LLaVA [4] | Video Chat2 [3] | Avg |
| OE | 33.32 ± 0.46 | 42.20 | 49.40 | 44.65 | 49.20 | 41.70 | 44.65 | 45.90 | 50.00 | 45.96 |
| OR | 24.96 ± 0.19 | 33.62 | 51.61 | 24.31 | 60.23 | 25.06 | 43.25 | 40.55 | 86.75 | 45.67 |
| AR | 25.26 ± 0.17 | 27.00 | 32.54 | 24.28 | 35.34 | 24.56 | 32.98 | 31.71 | 66.27 | 34.34 |
| AS | 26.12 ± 1.23 | 25.86 | 32.76 | 22.41 | 29.74 | 27.16 | 33.19 | 25.86 | 54.31 | 31.41 |
| AP | 24.16 ± 1.31 | 25.48 | 27.88 | 24.52 | 29.81 | 28.37 | 27.88 | 28.37 | 50.00 | 30.29 |
| AL | 25.49 ± 0.39 | 24.49 | 23.25 | 21.22 | 24.67 | 25.45 | 24.14 | 24.32 | 21.22 | 23.60 |
| OC | 25.17 ± 1.18 | 24.14 | 27.59 | 24.71 | 26.44 | 25.29 | 31.61 | 31.03 | 64.94 | 31.97 |
| AC | 25.06 ± 0.37 | 24.43 | 25.51 | 22.92 | 24.99 | 23.78 | 24.34 | 22.49 | 29.16 | 24.70 |
| RS | 19.46 ± 0.71 | 22.38 | 27.07 | 25.69 | 35.08 | 22.65 | 36.46 | 21.27 | 68.23 | 32.35 |
| | Special Task | | | | | | | | | |
| PM | 33.58 ± 0.41 | 43.19 | 48.00 | 44.88 | 50.68 | 40.28 | 49.70 | 45.37 | 52.44 | 46.82 |
| BM | 33.63 ± 1.21 | 39.31 | 50.29 | 43.35 | 47.98 | 44.80 | 48.84 | 42.20 | 47.69 | 45.56 |
| SA | 33.22 ± 0.54 | 41.08 | 48.87 | 47.23 | 49.47 | 42.96 | 48.18 | 44.16 | 52.42 | 46.80 |
| PD | 33.15 ± 1.55 | 40.56 | 47.55 | 46.85 | 50.35 | 38.46 | 44.06 | 45.80 | 54.20 | 45.98 |
| | Overall Performance | | | | | | | | | |
| Avg | 27.89 ± 2.90 | 31.83 | 37.87 | 32.08 | 39.54 | 31.58 | 37.64 | 34.54 | 53.66 | 37.34 |

Table 2: The evaluation results of 8 multimodal video models on our Animal-Bench (the first place for each task is marked in red, and the second place is marked in blue, and those below random accuracy are marked in gray).

**Effectiveness evaluation results** The table 2 presents the evaluation results of eight existing multimodal video models on our Animal-Bench. As models evolve, their performance also improves. The recently released model VideoChat2 [3] surpasses previous methods in most tasks. Notably, VideoChat2 achieves an accuracy of 86.75% in object recognition tasks and 68.23% in reasoning tasks. However, we also observe shortcomings in existing models on certain tasks. For instance, in action localization and action counting tasks, the answers provided by existing models are nearly equivalent to random guesses. These tasks typically require strong temporal understanding capabilities, which cannot be inferred solely from spatial scene comprehension. This indicates a need for enhancement in the temporal modeling abilities of existing models. Additionally, in the object existence task, the models tend to respond with "yes", indicating a severe hallucination problem.

**Robustness evaluation results**    We test the robustness of our models and their sensitivity to different types of variations on four types of simulated real-world data. We select the top four models in terms of effectiveness evaluation for robustness testing, as models with lower accuracy tend to provide responses close to random guessing, rendering discussions about robustness less meaningful. As shown in Table 2, VideoChat2 demonstrates relatively good robustness, with an overall decrease in accuracy of 3.70%. However, Video-LLaMA [8] shows sensitivity to the four simulated variations, with an overall decrease in accuracy of 8.72%. As depicted in Figure 5, we calculated the average accuracy decrease of the models for the four types of variations, revealing that models are more sensitive to shooting parameters than to changes in weather changes.

| Models | Weather condition | | Shooting parameters | | Overall |
|---|---|---|---|---|---|
| | Snow | Frost | Distance | Direction | |
| VideoChat2 | 1.49 | 2.17 | 4.76 | 6.39 | 3.70 |
| Video-LLaMA | 5.41 | 7.46 | 10.82 | 11.19 | 8.72 |
| VideoChat | 7.43 | 4.41 | 5.22 | 8.63 | 6.42 |
| Chat-UniVi | 5.04 | 1.81 | 3.83 | 7.86 | 4.64 |

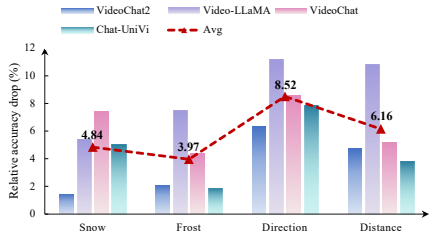

Table 3: Sensitivity of multimodal video models to different variations(relative accuracy drop(%)).

Figure 5: Average decrease in model accuracy(%) across four types of variations.

## 4.3   Further discussion

**Video editing samples**    As shown in Figure 6, we demonstrate the effects of our video editing pipeline used for simulating shooting parameters. It can be observed that the guided frames obtained through spectrum filtering have initially achieved good results. However, there still exist unnatural transitions at the edges of the original image. After undergoing the diffusion noise addition process again, the edges of the original image can transition better into the newly generated parts of the image. Since we employed a frozen stable diffusion model, the generation effect relies on the performance of this model.

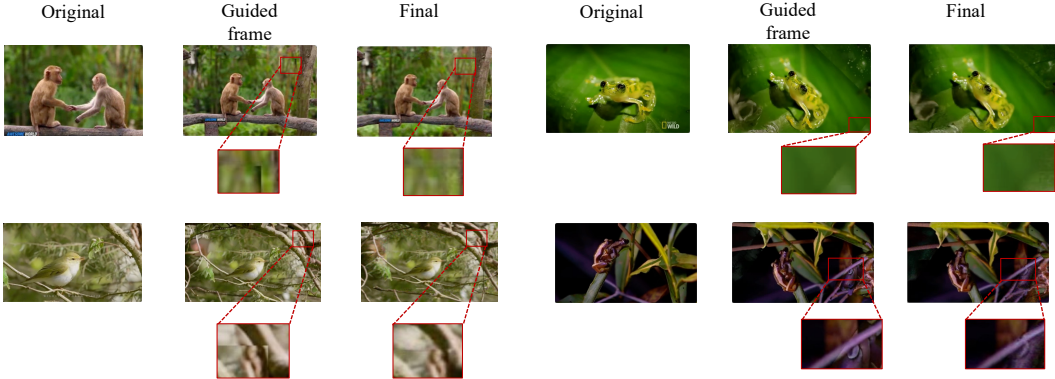

Figure 6: The visualization results of simulated changes in shooting parameters. Zoom in to view details.

**Animal category bias**    We aim to investigate whether the models exhibit biases towards different animal categories. We analyze the performance of multimodal video models in two tasks: object recognition and action recognition. We choose these two tasks for analysis due to their involvement with the most diverse range of animal species. As shown in Figure 7, in the object recognition task, models demonstrate higher recognition accuracy for the "mammal" and "bird" categories, while accuracy is generally lower for categories like "amphibian" and "reptile". Similarly, in the action recognition task, models exhibit higher accuracy in recognizing "fish" and "mammal" categories.

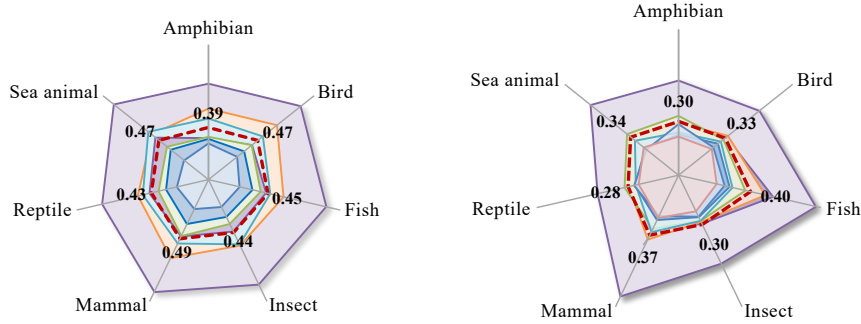

Figure 7: The radar map illustrates the accuracy on "object recognition" and "action recognition" tasks across 7 animal categories.

This could possibly be attributed to the greater prevalence of mammalian and fish species in the pretraining video datasets. Conversely, other categories suffer from lower recognition performance due to larger domain disparities.

**Model structure** We also analyze on the impact of model architecture on accuracy, we find from the data in Figure 8 that using the frozen CLIP ViT/L-14 [66] as the video encoder resulted in overall performance inferior to models employing larger or fine-tuned video encoders. This suggests that employing more powerful video encoders aids in a more comprehensive exploration of video features, which is of significant importance for the development of multi-modal video models. In addition, we observe that introducing a temporal modeling module into the model architecture is not particularly effective. According to Figure 8, models with an additional temporal modeling module outperform some earlier models in terms of answer accuracy but do not reach the level of some recently released models. This finding suggests that in the model design, the impact of the temporal modeling module may not be as significant as expected.

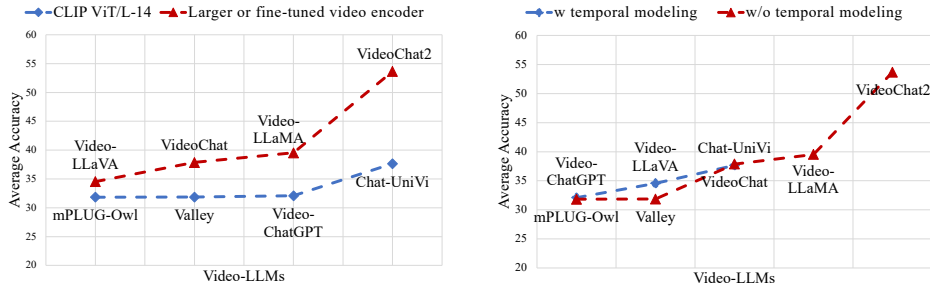

Figure 8: The line graph depicting the impact of video encoders and temporal modeling on the results.

## 5   Conclusion

In this work, we introduced Animal-Bench, an animal-centric benchmark to evaluate multimodal video models in real-world contexts. It includes 13 tasks across 7 major animal categories and 819 animal species. We proposed an automated pipeline of data processing. And we used video editing approaches to simulate realistic scenarios. We evaluated 8 current models on our Animal-Bench, and found significant room for improvement. We analyzed the bias of models towards animal categories and examined the influence of model architecture on experimental results. Our work aims to provide valuable insights and foster new research directions in multimodal video models.

## Acknowledgements

This work was supported in part by National Key R&D Program of China (2022ZD0116300), and in part by National Natural Science Foundation of China (NSFC) No. 62106022, 62225601, U23B2052, and in part by the Youth Innovative Research Team of BUPT No. 2023YQTD02, and in part by the Beijing Natural Science Foundation Project No. L242025, and in part by BUPT Excellent Ph.D. Students Foundation No. CX20241086, and in part by scholarships from China Scholarship Council (CSC) under Grant CSC No. 202406470082 and sponsored by Beijing Nova Program.

## References

[1] Bo Li, Yuanhan Zhang, Liangyu Chen, Jinghao Wang, Jingkang Yang, and Ziwei Liu. Otter: A multi-modal model with in-context instruction tuning.

[2] Feilong Chen, Minglun Han, Haozhi Zhao, Qingyang Zhang, Jing Shi, Shuang Xu, and Bo Xu. X-llm: Bootstrapping advanced large language models by treating multi-modalities as foreign languages. *arXiv preprint arXiv:2305.04160*, 2023.

[3] Kunchang Li, Yali Wang, Yinan He, Yizhuo Li, Yi Wang, Yi Liu, Zun Wang, Jilan Xu, Guo Chen, Ping Luo, et al. Mvbench: A comprehensive multi-modal video understanding benchmark. *arXiv preprint arXiv:2311.17005*, 2023.

[4] Bin Lin, Bin Zhu, Yang Ye, Munan Ning, Peng Jin, and Li Yuan. Video-llava: Learning united visual representation by alignment before projection. *arXiv preprint arXiv:2311.10122*, 2023.

[5] Yixuan Su, Tian Lan, Huayang Li, Jialu Xu, Yan Wang, and Deng Cai. Pandagpt: One model to instruction-follow them all. *arXiv preprint arXiv:2305.16355*, 2023.

[6] Junke Wang, Dongdong Chen, Chong Luo, Xiyang Dai, Lu Yuan, Zuxuan Wu, and Yu-Gang Jiang. Chatvideo: A tracklet-centric multimodal and versatile video understanding system. *arXiv preprint arXiv:2304.14407*, 2023.

[7] Junke Wang, Dongdong Chen, Zuxuan Wu, Chong Luo, Luowei Zhou, Yucheng Zhao, Yujia Xie, Ce Liu, Yu-Gang Jiang, and Lu Yuan. Omnivl: One foundation model for image-language and video-language tasks. *Advances in neural information processing systems*, 35:5696–5710, 2022.

[8] Hang Zhang, Xin Li, and Lidong Bing. Video-llama: An instruction-tuned audio-visual language model for video understanding. *arXiv preprint arXiv:2306.02858*, 2023.

[9] Bohao Li, Rui Wang, Guangzhi Wang, Yuying Ge, Yixiao Ge, and Ying Shan. Seed-bench: Benchmarking multimodal llms with generative comprehension. *arXiv preprint arXiv:2307.16125*, 2023.

[10] Yuan Liu, Haodong Duan, Yuanhan Zhang, Bo Li, Songyang Zhang, Wangbo Zhao, Yike Yuan, Jiaqi Wang, Conghui He, Ziwei Liu, et al. Mmbench: Is your multi-modal model an all-around player? *arXiv preprint arXiv:2307.06281*, 2023.

[11] Munan Ning, Bin Zhu, Yujia Xie, Bin Lin, Jiaxi Cui, Lu Yuan, Dongdong Chen, and Li Yuan. Video-bench: A comprehensive benchmark and toolkit for evaluating video-based large language models. *arXiv preprint arXiv:2311.16103*, 2023.

[12] Peng Xu, Wenqi Shao, Kaipeng Zhang, Peng Gao, Shuo Liu, Meng Lei, Fanqing Meng, Siyuan Huang, Yu Qiao, and Ping Luo. Lvlm-ehub: A comprehensive evaluation benchmark for large vision-language models. *arXiv preprint arXiv:2306.09265*, 2023.

[13] Weihao Yu, Zhengyuan Yang, Linjie Li, Jianfeng Wang, Kevin Lin, Zicheng Liu, Xinchao Wang, and Lijuan Wang. Mm-vet: Evaluating large multimodal models for integrated capabilities. *arXiv preprint arXiv:2308.02490*, 2023.

[14] Norman L Christensen, Ann M Bartuska, James H Brown, Stephen Carpenter, Carla d'Antonio, Rober Francis, Jerry F Franklin, James A MacMahon, Reed F Noss, David J Parsons, et al. The report of the ecological society of america committee on the scientific basis for ecosystem management. *Ecological applications*, 6(3):665–691, 1996.

[15] José M Gómez, Jordi Bosch, Francisco Perfectti, Juande Fernández, and Mohamed Abdelaziz. Pollinator diversity affects plant reproduction and recruitment: the tradeoffs of generalization. *Oecologia*, 153:597–605, 2007.

[16] Rafał Zwolak and Andrew Sih. Animal personalities and seed dispersal: a conceptual review. *Functional Ecology*, 34(7):1294–1310, 2020.

[17] Robert T Paine. Food web complexity and species diversity. *The American Naturalist*, 100(910):65–75, 1966.

[18] Yinuo Jing, Chunyu Wang, Ruxu Zhang, Kongming Liang, and Zhanyu Ma. Category-specific prompts for animal action recognition with pretrained vision-language models. In *Proceedings of the 31st ACM International Conference on Multimedia*, pages 5716–5724, 2023.

[19] David J Anderson and Pietro Perona. Toward a science of computational ethology. *Neuron*, 84(1):18–31, 2014.

[20] Liqi Feng, Yaqin Zhao, Yichao Sun, Wenxuan Zhao, and Jiaxi Tang. Action recognition using a spatial-temporal network for wild felines. *Animals*, 11(2):485, 2021.

[21] Jacob M Graving, Daniel Chae, Hemal Naik, Liang Li, Benjamin Koger, Blair R Costelloe, and Iain D Couzin. Deepposekit, a software toolkit for fast and robust animal pose estimation using deep learning. *Elife*, 8:e47994, 2019.

[22] Hung Nguyen, Sarah J Maclagan, Tu Dinh Nguyen, Thin Nguyen, Paul Flemons, Kylie Andrews, Euan G Ritchie, and Dinh Phung. Animal recognition and identification with deep convolutional neural networks for automated wildlife monitoring. In *2017 IEEE Int'l Conf. on Data Science and Advanced Analytics (DSAA)*, pages 40–49, 2017.

[23] Abhineet Singh, Marcin Pietrasik, Gabriell Natha, Nehla Ghouaiel, Ken Brizel, and Nilanjan Ray. Animal detection in man-made environments. In *Proceedings of the IEEE Winter Conf. on Applications of Computer Vision*, pages 1438–1449, 2020.

[24] Lukas von Ziegler, Oliver Sturman, and Johannes Bohacek. Big behavior: challenges and opportunities in a new era of deep behavior profiling. *Neuropsychopharmacology*, 46(1):33–44, 2021.

[25] Hüseyin Gökhan Akçay, Bekir Kabasakal, Duygugül Aksu, Nusret Demir, Melih Öz, and Ali Erdoğan. Automated bird counting with deep learning for regional bird distribution mapping. *Animals*, 10(7):1207, 2020.

[26] Dominique Chabot and Charles M Francis. Computer-automated bird detection and counts in high-resolution aerial images: a review. *Journal of Field Ornithology*, 87(4):343–359, 2016.

[27] Arturo Gomez Chavez, Jorge Fontes, Pedro Afonso, and Max Pfingsthorn. Automated species counting using a hierarchical classification approach. In *IEEE Oceans*.

[28] Marcelo Feighelstein, Yamit Ehrlich, Li Naftaly, Miriam Alpin, Shenhav Nadir, Ilan Shimshoni, Renata H Pinho, Stelio PL Luna, and Anna Zamansky. Deep learning for video-based automated pain recognition in rabbits. *Scientific Reports*, 13(1):14679, 2023.

[29] Marcelo Feighelstein, Ilan Shimshoni, Lauren R Finka, Stelio PL Luna, Daniel S Mills, and Anna Zamansky. Automated recognition of pain in cats. *Scientific Reports*, 12(1):9575, 2022.

[30] Krista McLennan and Marwa Mahmoud. Development of an automated pain facial expression detection system for sheep (ovis aries). *Animals*, 9(4):196, 2019.

[31] Yamin Han, Jie Wu, Hongming Zhang, Mingyu Cai, Yang Sun, Bin Li, Xilong Feng, Jinye Hao, and Hanchen Wang. Beef cattle abnormal behaviour recognition based on dual-branch frequency channel temporal excitation and aggregation. *Biosystems Engineering*, 241:28–42, 2024.

[32] Xin Li, Yinfeng Hao, Muhammad Akhter, Daoliang Li, et al. A novel automatic detection method for abnormal behavior of single fish using image fusion. *Computers and Electronics in Agriculture*, 203:107435, 2022.

[33] Melinda A Novak and Jerrold S Meyer. Abnormal behavior of animals in research settings. In *Behavioral Biology of Laboratory Animals*, pages 27–50. CRC Press, 2021.

[34] Xiaodan Li, Yuefeng Chen, Yao Zhu, Shuhui Wang, Rong Zhang, and Hui Xue. Imagenet-e: Benchmarking neural network robustness via attribute editing. In *Proceedings of the IEEE/CVF Conference on Computer Vision and Pattern Recognition*, pages 20371–20381, 2023.

[35] Robin Rombach, Andreas Blattmann, Dominik Lorenz, Patrick Esser, and Björn Ommer. High-resolution image synthesis with latent diffusion models. In *Proceedings of the IEEE/CVF conference on computer vision and pattern recognition*, pages 10684–10695, 2022.

[36] Chaoyou Fu, Yuhan Dai, Yondong Luo, Lei Li, Shuhuai Ren, Renrui Zhang, Zihan Wang, Chenyu Zhou, Yunhang Shen, Mengdan Zhang, et al. Video-mme: The first-ever comprehensive evaluation benchmark of multi-modal llms in video analysis. *arXiv preprint arXiv:2405.21075*, 2024.

[37] Xun Long Ng, Kian Eng Ong, Qichen Zheng, Yun Ni, Si Yong Yeo, and Jun Liu. Animal kingdom: A large and diverse dataset for animal behavior understanding. In *Proceedings of the IEEE/CVF Conference on Computer Vision and Pattern Recognition*, pages 19023–19034, 2022.

[38] Jun Chen, Ming Hu, Darren J Coker, Michael L Berumen, Blair Costelloe, Sara Beery, Anna Rohrbach, and Mohamed Elhoseiny. Mammalnet: A large-scale video benchmark for mammal recognition and behavior understanding. In *Proceedings of the IEEE/CVF Conference on Computer Vision and Pattern Recognition*, pages 13052–13061, 2023.

[39] Will Kay, Joao Carreira, Karen Simonyan, Brian Zhang, Chloe Hillier, Sudheendra Vijaya-narasimhan, Fabio Viola, Tim Green, Trevor Back, Paul Natsev, et al. The kinetics human action video dataset. *arXiv preprint arXiv:1705.06950*, 2017.

[40] Junbin Xiao, Xindi Shang, Angela Yao, and Tat-Seng Chua. Next-qa: Next phase of question-answering to explaining temporal actions. In *Proceedings of the IEEE/CVF conference on computer vision and pattern recognition*, pages 9777–9786, 2021.

[41] Raghav Goyal, Samira Ebrahimi Kahou, Vincent Michalski, Joanna Materzynska, Susanne Westphal, Heuna Kim, Valentin Haenel, Ingo Fruend, Peter Yianilos, Moritz Mueller-Freitag, et al. The" something something" video database for learning and evaluating visual common sense. In *Proceedings of the IEEE international conference on computer vision*, pages 5842–5850, 2017.

[42] Dejing Xu, Zhou Zhao, Jun Xiao, Fei Wu, Hanwang Zhang, Xiangnan He, and Yueting Zhuang. Video question answering via gradually refined attention over appearance and motion. In *Proceedings of the 25th ACM international conference on Multimedia*, pages 1645–1653, 2017.

[43] Jun Xu, Tao Mei, Ting Yao, and Yong Rui. Msr-vtt: A large video description dataset for bridging video and language. In *Proceedings of the IEEE conference on computer vision and pattern recognition*, pages 5288–5296, 2016.

[44] Qinghao Ye, Haiyang Xu, Guohai Xu, Jiabo Ye, Ming Yan, Yiyang Zhou, Junyang Wang, Anwen Hu, Pengcheng Shi, Yaya Shi, et al. mplug-owl: Modularization empowers large language models with multimodality. *arXiv preprint arXiv:2304.14178*, 2023.

[45] Viorica Patraucean, Lucas Smaira, Ankush Gupta, Adria Recasens, Larisa Markeeva, Dylan Banarse, Skanda Koppula, Mateusz Malinowski, Yi Yang, Carl Doersch, et al. Perception test: A diagnostic benchmark for multimodal video models. *Advances in Neural Information Processing Systems*, 36, 2024.

[46] Viraj Prabhu, Sriram Yenamandra, Prithvijit Chattopadhyay, and Judy Hoffman. Lance: Stress-testing visual models by generating language-guided counterfactual images. *Advances in Neural Information Processing Systems*, 36, 2024.

[47] Dan Hendrycks and Thomas Dietterich. Benchmarking neural network robustness to common corruptions and perturbations. *arXiv preprint arXiv:1903.12261*, 2019.

[48] Shresth Grover, Vibhav Vineet, and Yogesh Rawat. Revealing the unseen: Benchmarking video action recognition under occlusion. *Advances in Neural Information Processing Systems*, 36, 2024.

[49] Madeline Chantry Schiappa, Naman Biyani, Prudvi Kamtam, Shruti Vyas, Hamid Palangi, Vibhav Vineet, and Yogesh S Rawat. A large-scale robustness analysis of video action recognition models. In *Proceedings of the IEEE/CVF Conference on Computer Vision and Pattern Recognition*, pages 14698–14708, 2023.

[50] John A Endler. *Natural selection in the wild*. Number 21. Princeton University Press, 1986.

[51] Edward O Wilson. *Sociobiology: The new synthesis*. Harvard University Press, 2000.

[52] Robert L Trivers. Parental investment and sexual selection. In *Sexual selection and the descent of man*, pages 136–179. Routledge, 2017.

[53] Robert M Liptrap. Stress and reproduction in domestic animals. *Annals of the New York Academy of Sciences*, 697(1):275–284, 1993.

[54] Jie Lei, Licheng Yu, Mohit Bansal, and Tamara L Berg. Tvqa: Localized, compositional video question answering. *arXiv preprint arXiv:1809.01696*, 2018.

[55] Makarand Tapaswi, Yukun Zhu, Rainer Stiefelhagen, Antonio Torralba, Raquel Urtasun, and Sanja Fidler. Movieqa: Understanding stories in movies through question-answering. In *Proceedings of the IEEE conference on computer vision and pattern recognition*, pages 4631–4640, 2016.

[56] Kexin Yi, Chuang Gan, Yunzhu Li, Pushmeet Kohli, Jiajun Wu, Antonio Torralba, and Joshua B Tenenbaum. Clevrer: Collision events for video representation and reasoning. *arXiv preprint arXiv:1910.01442*, 2019.

[57] Yunseok Jang, Yale Song, Youngjae Yu, Youngjin Kim, and Gunhee Kim. Tgif-qa: Toward spatio-temporal reasoning in visual question answering. In *Proceedings of the IEEE conference on computer vision and pattern recognition*, pages 2758–2766, 2017.

[58] Dan Liu, Jin Hou, Shaoli Huang, Jing Liu, Yuxin He, Bochuan Zheng, Jifeng Ning, and Jingdong Zhang. Lote-animal: A long time-span dataset for endangered animal behavior understanding. In *Proceedings of the IEEE/CVF International Conference on Computer Vision*, pages 20064–20075, 2023.

[59] OpenAI. Chatgpt. *https://openai.com/blog/chatgpt/*, 2023.

[60] Jonathan Ho, Ajay Jain, and Pieter Abbeel. Denoising diffusion probabilistic models. *Advances in neural information processing systems*, 33:6840–6851, 2020.

[61] Diederik P Kingma and Max Welling. Auto-encoding variational bayes. *arXiv preprint arXiv:1312.6114*, 2013.

[62] KunChang Li, Yinan He, Yi Wang, Yizhuo Li, Wenhai Wang, Ping Luo, Yali Wang, Limin Wang, and Yu Qiao. Videochat: Chat-centric video understanding. *arXiv preprint arXiv:2305.06355*, 2023.

[63] Muhammad Maaz, Hanoona Rasheed, Salman Khan, and Fahad Shahbaz Khan. Video-chatgpt: Towards detailed video understanding via large vision and language models. *arXiv preprint arXiv:2306.05424*, 2023.

[64] Ruipu Luo, Ziwang Zhao, Min Yang, Junwei Dong, Da Li, Pengcheng Lu, Tao Wang, Linmei Hu, Minghui Qiu, and Zhongyu Wei. Valley: Video assistant with large language model enhanced ability. *arXiv preprint arXiv:2306.07207*, 2023.

[65] Peng Jin, Ryuichi Takanobu, Caiwan Zhang, Xiaochun Cao, and Li Yuan. Chat-univi: Unified visual representation empowers large language models with image and video understanding. *arXiv preprint arXiv:2311.08046*, 2023.

[66] Alec Radford, Jong Wook Kim, Chris Hallacy, Aditya Ramesh, Gabriel Goh, Sandhini Agarwal, Girish Sastry, Amanda Askell, Pamela Mishkin, Jack Clark, et al. Learning transferable visual models from natural language supervision. In *International conference on machine learning*, pages 8748–8763. PMLR, 2021.

[67] Del Thiessen and Maureen Rice. Mammalian scent gland marking and social behavior. *Psychological bulletin*, 83(4):505, 1976.

# Appendix

## A  More Details on QA Generation

In Table 5, we provide a detailed breakdown of task divisions from coarse-grained to fine-grained levels, specifying the data used for each task. We have established specific rules for generating QA pairs based on the data requirements of different tasks. For tasks without existing QA pairs, we use ChatGPT [59] to automatically generate three questions and randomly select one to minimize model bias towards the questions. We also present the amount of data included for each task. Except for a few tasks, most tasks have sufficient data.

## B  Option Difficulty

Quantifying the difficulty of options is inherently challenging. In this study, we employ a qualitative analysis approach to achieve a moderate level of difficulty for the options. For the action recognition task, we examined the frequency of various actions and found that they adhere to a long-tail distribution. We categorize common actions, or "head actions," such as "running" and "eating," as simple options that can be identified without specialized knowledge. In contrast, rare actions, or "tail actions," such as "molting" in birds, require specialized knowledge to identify and are thus classified as difficult options. Our approach involves incorporating the correct answer along with two simple options and one difficult option, thereby ensuring that the difficulty of the options is balanced and reflective of the natural frequency distribution of actions. For the object recognition task, we test four situations:

- Random selection: Besides the correct answer, the other three options are randomly selected from all the animal species involved.

- Different major categories: Besides the correct answer, the other three options are randomly selected from different major animal categories than the correct answer. This setting makes the question-answer pairs easier because it is a coarse-grained judgment. The difference between the other three options and the correct answer is large, and if the model can identify correctly at the coarse-grained level, it can answer correctly.

- Same major category: Besides the correct answer, the other three options are randomly selected from the same major animal category as the correct answer. This setting makes the question-answer pairs more difficult because it is a fine-grained judgment. The difference between the other three options and the correct answer is small.

- Rules designed in this paper: Besides the correct answer, two options come from a different major animal category than the correct answer, and one option come from the same major animal category as the correct answer. This design makes the question-answer pairs neither too difficult nor too easy.

The comparative experimental results regarding the four rules are shown in Table 4. The results indicate that the selection of options affects the experiment results, which also supports our theoretical analysis above. Our design can moderate the difficulty of the question-answer pairs, making the evaluation of the model more aligned with real-world scenarios.

| Rules | Acc (%) |
| --- | --- |
| Different major categories | 97.31 |
| Random selection | 91.96 |
| Same major category | 76.34 |
| Rules designed in the paper | 86.75 |

Table 4: The table of accuracy rates of VideoChat2's responses for the four different option generation rules.

| | Ability | Coarse task | Fine-grained task | Datasets | Rules | Size |
|---|---|---|---|---|---|---|
| **Common Task** | **Perception** | **Object** | **Object Existence** | MammalNet | Q: Is there a/an (X) in this video? (X): 50% probability chosen from the GT labels 25% probability chosen from the other animals with the same mian class as GT 25% probability chosen from other classes compared with GT O: "yes", "no", "not sure"(shuffle) | 2000 |
| | | | **Object Recognition** | MammalNet Animal Kingdom | Q: Randomly select three sentence patterns generated by ChatGPT O: One option from the GT label One option from the the other animas with the same main class as GT Two options from other main classes compared with GT | 15000 |
| | | **Action** | **Action Recognition** | MammalNet Animal Kingdom | Data: (1) Choose data that are with single action label annotation (2) 3 < Duration < 35 Q: Randomly choose one out of three ChatGPT-generated questions O: One option from the GT label Two options from the top 50% most frequent actions One option from the least frequent 50% actions | 15060 |
| | | | **Action Sequence** | Animal Kingdom(VG) | Data: (1) Choose data which include only one kind of animal with more than two different actions (2) 3 < Duration < 35 Q: Sentence pattern generated by ChatGPT O: One option from the GT label Two options from the top 50% most frequent actions One option from the least frequent 50% actions | 232 |
| | | | **Action Prediction** | Animal Kingdom(VG) | Data: (1) Choose data which include only one kind of animal with more than two different actions (2) 3 < Duration < 35 Q: Sentence pattern generated by ChatGPT O: One option from the GT label Two options from the top 50% most frequent actions One option from the least frequent 50% actions | 208 |
| | | **Time** | **Action Localization** | Animal Kingdom(VG) | Data: (1) Choose videos that do not contain repeated actions. (2) Split the video into three time zones (beginning/middle/ end/throughout). Next, re-annotate the data into those labels by checking in which time zone the action lasts the longest. If the grounding occurs at the very beginning or end of the video and lasts longer than 50% of the total video length, it should be removed. (3) 3 < Duration < 35 Q: Randomly choose one out of three ChatGPT-generated questions O: "In the middle of the video." "Throughout the entire video." "At the end of the video.", "At the beginning of the video." (Shuffle) | 1682 |
| | | **Count** | **Object Count** | MSRVTT-QA | Data: 3 < Duration < 35 Q, C: Utilize ChatGPT to adopt those animal-centric videos and annotations in MSRVTT-QA | 174 |
| | | | **Action Count** | TGIF-QA | Data: 3 < Duration < 35 Q, C: Utilize ChatGPT to adopt those animal-centric videos and annotations in TGIF-QA | 2325 |
| | **Cognition** | **Reasoning** | **Abductive Reasoning** | NExT-QA | Data: 3 < Duration < 35 Q, C: Utilize ChatGPT to adopt those animal-centric videos and annotations in NExT-QA | 362 |
| **Special Task** | **Perception** | **Special Behavior Detection** | **Predator-Prey Behavior Detection** | MammalNet Animal Kingdom | Data: (1) Choose data that include action of "hunts other animals","camouflaging" etc. (2) 3 < Duration < 35 Q: Sentence pattern generated by ChatGPT C: "yes", "no", "not sure" (shuffle) | 1827 |
| | | | **Social Interaction Analysis** | MammalNet Animal Kingdom LoTE-Annimal | Data: (1) Choose data that include action of "hugging","circumanal gland signing", "urine signing" and "aggregation" etc. (2) 3 < Duration < 35 Q: Sentence pattern generated by ChatGPT C: "yes", "no", "not sure" (shuffle) | 2337 |
| | | | **Breeding Behavior Monitoring** | MammalNet Animal Kingdom LoTE-Annimal | Data: (1) Choose data that include action of "parental", "mates with other animals", "gives birth to a baby" and "nurses or breastfeeds its baby" etc. (2) 3 < Duration < 35 Q: Sentence pattern generated by ChatGPT C: "yes", "no", "not sure" (shuffle) | 346 |
| | | | **Stress and Pain Detection** | MammalNet Animal Kingdom | Data: (1) Choose data that include action of "vomits" and "dying" (2) 3 < Duration < 35 Q: Sentence pattern generated by ChatGPT C: "yes", "no", "not sure" (shuffle) | 286 |

Table 5: More details about how Animal-Bench divide tasks, filter data and process data into QA format.

## C    More Experiment Details

As shown in Table 6, we standardized the parameters of all language models used to ensure the fairness of the experiments. Following [3], we designed a uniform system prompt and answer prompt. By setting the answer prompt, our model outputs correspond to one of the options. In robustness testing, we selected 100 data samples from each task in the original Animal-Bench. Our selection process adhered to two principles: randomness, to minimize bias introduced by human intervention; and diversity, ensuring that the selected data represents the species in the original dataset. We believe that these data samples are sufficient to represent our original dataset and can reduce computational burden. We set different levels of severity for each variation. For weather changes, we set five variation parameters and randomly selected one for each piece of data. For size and angle changes, to simulate distant scenes, we set the ratio of the simulated video height to the original video height between $0.1 * (3 - 7)$; to simulate close scenes, we set the scale between $0.1 * (1.3 - 1.7)$. To simulate direction changes, we set the angle between $(30° - 90°)$ and $(270° - 330°)$ to mimic realistic conditions.

| Parameters | Value |
|---|---|
| System Prompt | Carefully watch the video and pay attention to the cause and sequence of events, the detail and movement of objects, and the action and pose of animals. Based on your observations, select the best option that accurately addresses the question. |
| Question Prompt | Only give the best option. |
| Answer Prompt | Best option: ( |
| Crop Size | $(224, 224)$ |
| Number of Frames | 16 |
| Max Tokens | 200 |
| Temperature | 1.0 |
| Scale | $0.1 * (3 - 7), 0.1 * (13 - 17)$ |
| Angle | $(30° - 90°), (270° - 330°)$ |

Table 6: Detailed Experimental Parameters.

## D    Ablation Study on Data Preprocessing

We conduct an ablation study on the number of video frames input to the model and the frame cropping method. We test with both 16 frames and 8 frames, designing two different video frame cropping settings.

Setting 1: The specific video preprocessing process is as follows: if $H > W$, the frame is scaled to $(224, 224 \times H/W)$. If $W > H$, the frame is scaled to $(224 \times W/H, 224)$. After scaling, the video frames are center-cropped to obtain a center region of $(224, 224)$.

Setting 2: We also experiment with padding non-square videos along the shorter side to make them square before scaling them to $(224, 224)$.

The experimental results are shown in Table 7. Reducing the number of frames leads to a decrease in the model's accuracy on most time-related tasks, which demonstrates that increasing the input video frames benefits the model in extracting temporal information and improving accuracy. However, in the object counting task, reducing the number of frames results in a significant performance improvement. We believe this is because the MSRVTT-QA dataset used for object counting contains video segments from different perspectives within a single video. These different perspectives may capture various states of instances, and increasing the number of frames could confuse the model when processing redundant information from different perspectives, making it difficult to distinguish between instances. We also find that using the padding-then-cropping video frame preprocessing method leads to a slight decrease in the model's accuracy.

| Methods | OE | OR | AR | AS | AP | AL | OC | AC | RS | PM | BM | SA | PD | Avg |
|---|---|---|---|---|---|---|---|---|---|---|---|---|---|---|
| num=16, Setting 1 | 50.00 | 86.75 | 66.27 | 54.31 | 50.00 | 21.22 | 64.94 | 29.16 | 68.23 | 52.40 | 47.69 | 52.42 | 54.20 | 53.66 |
| num=8, Setting 1 | 50.00 | 86.67↓0.08 | 65.88↓0.39 | 54.74↑0.43 | 46.15↓3.85 | 21.28↑0.06 | 69.54↑4.6 | 28.95↓0.21 | 67.40↓0.83 | 52.40 | 47.69 | 52.42 | 54.20 | 53.64↓0.02 |
| num=16, Setting 2 | 50.00 | 85.13↓1.62 | 64.72↓1.55 | 51.72↓2.59 | 50.96↑0.96 | 22.17↑0.95 | 62.64↓2.30 | 28.99↓0.17 | 66.57↓1.66 | 52.30↓0.01 | 47.69 | 52.42 | 54.20 | 53.04↓0.62 |

Table 7: Ablation study results table for data preprocessing.

# E Results on MSRVTT-QA and TGIF-QA

We also analyze the models' performance on all data from MSRVTT-QA and TGIF-QA, as well as the animal subset data from Animal-Bench, as shown in Table 8. The model performs better on the overall data but worse on the animal subsets. This indicates, to some extent, that the model exhibits agent bias, demonstrating a stronger understanding of videos with humans and objects as agents compared to those with animals as agents.

| Method | MSRVTT-QA | | TGIF-QA | |
|---|---|---|---|---|
| | Original All Data | Animal-Bench (Object Count) | Original All Data | Animal-Bench (Action Count) |
| VideoChat | 45.0 | 27.6 | 34.4 | 25.5 |
| Video-LLaMA | 29.6 | 26.4 | – | 25.0 |
| Video-ChatGPT | 49.3 | 24.7 | 51.4 | 22.9 |
| Video-LLaVA | 59.2 | 31.0 | 70.0 | 22.5 |

Table 8: Experimental results comparison on MSRVTT-QA, TGIF-QA, and Animal-Bench (object count and action count).

# F Multimodal Video Model Parameters

In Table 9, we present a comparison of the parameters of the multimodal video models used in our experiments. We provide a detailed description of the model parameters from aspects such as video encoder, temporal module, large language model, training datasets, and the size of the training datasets. By analyzing these alongside the experimental results, we obtain several discussions on the model structure in the main part of the paper. In addition to the discussions on the video encoder and temporal module mentioned in the main paper, we also find that the amount of training data does not significantly affect the experimental results; a larger training dataset does not necessarily lead to better model performance, and vice versa. We hope these findings on the impact of model structure and training process on experimental results can inspire future development of multimodal video models.

# G Data Statistics on Different Agents

In Fig. 9, we present statistics on the agents involved in each task of MVBench. It can be observed that most tasks only include human agents or objects interacting with humans, with very few tasks involving animal agents. This further illustrates that existing evaluation benchmarks are biased towards humans, and the evaluation of multimodal video models in the domain of animals is lacking.

We have calculated the occurrence frequency of each animal species in our Animal-Bench dataset, as depicted in Figure 10. It can be observed that the distribution of animals is not uniform, following a long-tail distribution, which aligns with the general pattern where some species are more abundant while others are less so in the natural world.

| MLLMs | Video Encoder | Temporal Module | LLM | Training Dataset | Training Dataset Size |
|---|---|---|---|---|---|
| mPLUG-Owl | CLIP ViT/L-14 | / | LLaMA-7B | stage 1: LAION-400, MCOYO-700M, Conceptual Captions, MSCOCO<br>stage 2: ALpaca, Vicuna, Baize, LLaVA | stage 1: 1105M<br>stage 2: 392K |
| Video-ChatGPT | CLIP ViT/L-14 | / | Vicuna-7B | video instruction pairs | 100K |
| Valley | CLIP ViT/L-14 | Temporal attention, Temporal token | Stable-Vicuna-7B | stage 1: CC595K, WebVid2M<br>stage 2: video, image based instruction data<br>(combined, LLaVA, Video-ChatGPT) | stage 1: 1297K<br>stage 2: 234K |
| VideoChat | ViT-G/14 | Global Multi-Head Relation Aggregator | Stable Vicuna-7B | stage 1: WebVid-10M, COCO, Caption, Visual Genome, SBU Captions, CC3M, CC12M<br>stage 2: part of MiniGPT-4, LLaVA | stage 1: 25M<br>stage 2: 18K |
| Chat-UniVi | CLIP ViT-L/14 | Dynamic visual token | Vicuna-7B | stage 1: image-caption pairs (COCO, CC595K)<br>stage 2: MiMIC-IT, LLaVA, Video-ChatGPT | stage 1: 595K<br>stage 2: — |
| Video-LLaMA | ViT-G/14 | / | Vicuna-7B | stage 1: Webvid-2M, CC595K<br>stage 2: MiniGPT-4, LLaVA, Video-ChatGPT | stage 1: 2M<br>stage 2: 649K |
| VideoChat2 | UMT-L | / | Vicuna-7B | stage 1: CC3M, CC12M, WebVid-10M<br>stage 2: COCO, Visual Genome, SBU, InternVid<br>stage 3: 6 categories | stage 1: 15M<br>stage 2: 12M<br>stage 3: 1.9M |
| Video-LLaVA | ViT-L/14 + LoRA finetune | / | Vicuna-7B | stage 1: LAION-CC-SBU, WebVid<br>stage 2: LLaVA, Video-ChatGPT | stage 1: 1260K<br>stage 2: 765K |

Table 9: Comparison of parameters of different multimodal video models, where "/" indicates that no trainable temporal modules are introduced.

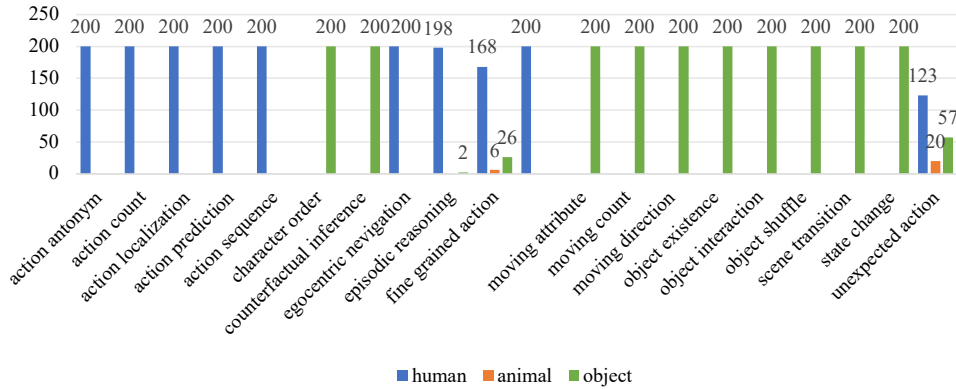

Figure 9: The data quantity for different agents in each task of MVBench.

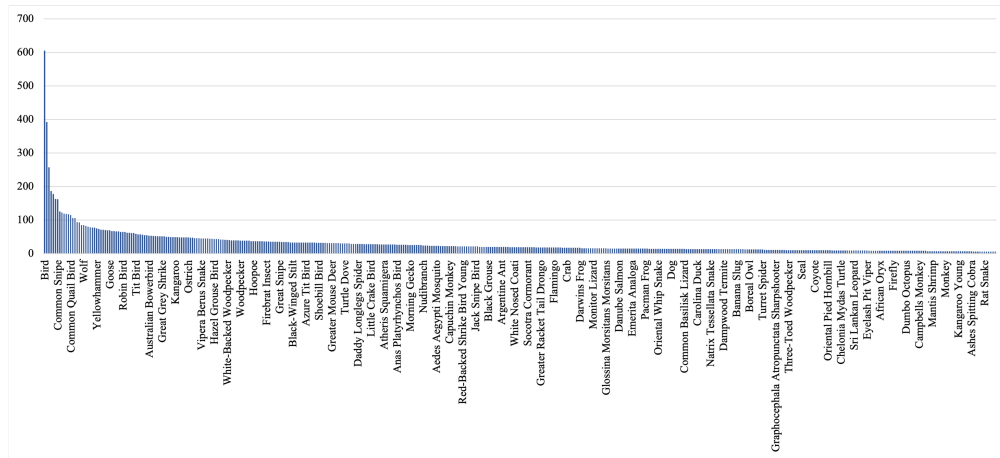

Figure 10: Statistical chart of data quantities for different animal species.

# H Potential Negative Impacts

Although the technology itself is intended to protect and study wildlife, if it falls into the hands of malicious actors, it could be used for illegal hunting, poaching, and animal exploitation. These actions could cause severe damage to wildlife populations and make conservation efforts more challenging. Also, excessive reliance on technology for animal monitoring and protection may lead to neglect of manual patrols and traditional conservation methods. If the technology fails or is compromised, it could result in serious security vulnerabilities. Given these potential negative impacts, it is recommended to implement strict security measures in technology application and data management, and to establish appropriate laws and regulations to ensure the technology is used correctly and safely.

# I Limitations and Future Directions

Our work has some limitations: First, the responses of multimodal video models may be influenced by the input parameters of the models, and different parameter settings could affect the accuracy of the responses. We did not test the sensitivity of the model to these parameters and instead used uniform parameters for the models. Second, since our editing is based on stable diffusion, the editing effect is subject to the capabilities and limitations of stable diffusion. Third, video editing may introduce some counterfactuals or additional animals that could change the answers to the questions. We have made efforts to minimize this occurrence through manual selection. In future work, we will continue to study the impact of model parameters on model performance and conduct deeper research on diffusion models to achieve better editing effects. We also plan to reduce the appearance of new animals in the future by adding negative prompts and conducting further exploration.

# J Assets and Licenses

In our paper, we use data from six datasets: Animal-Kingdom [37], MammalNet [38], LoTE-Animal [58], MSRVTT-QA [42], NExT-QA [40], and TGIF-QA [57]. We appreciate the contributions of the aforementioned works, all of which have been cited in the main article. Specifically:

- MammalNet is licensed under the CC BY license.
- LoTE-Animal is licensed under the Creative Commons Attribution-ShareAlike 4.0 International License.
- MSRVTT-QA is licensed under the MIT license.
- NExT-QA is licensed under the MIT license.
- For the Animal-Kingdom dataset, we have contacted the authors by filling out a questionnaire regarding the dataset's use and have obtained an official download link. Additionally, we have emailed the authors about our use of the dataset in our paper.
- The TGIF-QA dataset is explicitly stated on its GitHub page "to be free to use for academic purposes only."

We believe that our work fully respects the original authors' copyrights, and all assets have been used appropriately.

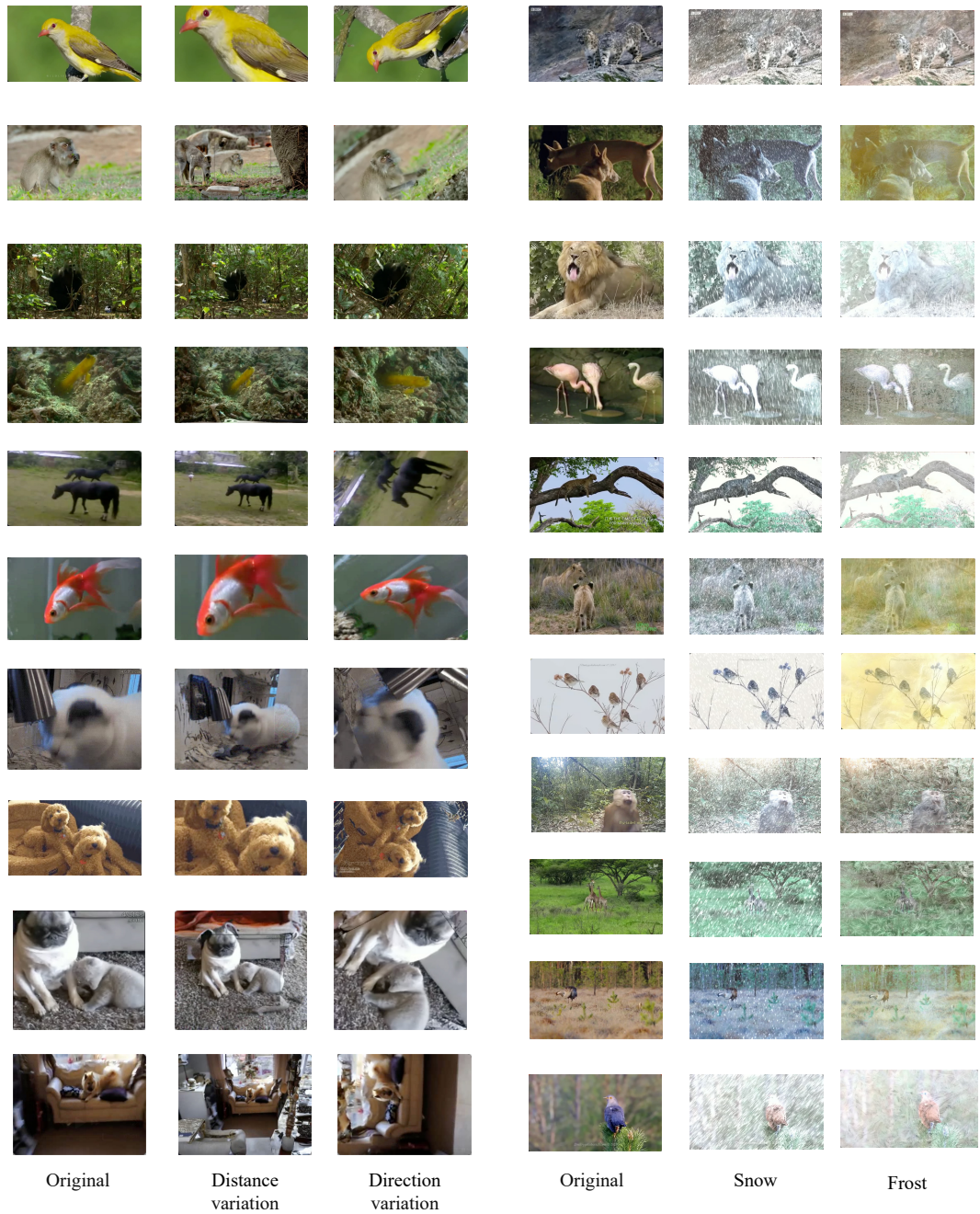

Original     Distance variation     Direction variation     Original     Snow     Frost

Figure 11: Visualization of simulated changes mimicking real shooting scenarios

**Object Recognition**

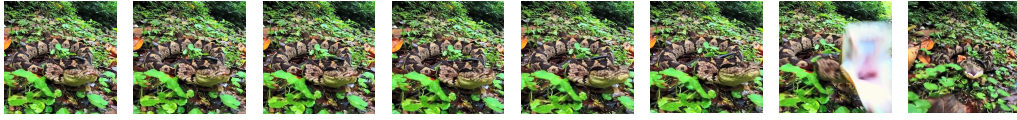

**Which animal can be seen in the video?**
(A)metlapilcoatlus mexicanus jumping pit viper
(B)monkey young
(C)stoat
(D)thamnophis cyrtopsis snake

Right answer: (A)metlapilcoatlus mexicanus jumping pit viper

Chat-UniVi: (A) metlapilcoatlus mexicanus jumping pit viper.
mPLUG-Owl: (A) metlapilcoatlus mexicanus jumping pit viper\n\nThe video shows a metlapilcoatlus mexicanus jumping pit viper, which is also known as a Central American rattlesnake.
Valley: (A) metapilcopus mexicanus jumping pit viper
VideoChat: (A) metlapilcoatlus mexicanus jumping pit viper. Based on the video, the animal is a brown snake with black markings that is visible when it jumps up in the air and disappears into the vegetation. This snake is identified as the metlapilcoatlus mexicanus, a species of pit viper that is known for its jumping ability. It is found in tropical and subtropical regions of the Americas and is
VideoChat2: (D) thamnophis cyrtopsis snake
Video-ChatGPT: (A) metlapilcoatlus mexicanus jumping pit viper; The video shows a snake jumping and slithering in the trees near a tree stump. The scene shows the snake moving quickly away from the person looking at it, causing it to move into a different position and then hide under the tree.
Video-LLaMA: (D) thamnophis cyrtopsis snake.
Video-LLaVA: (A)

**Object Existence**

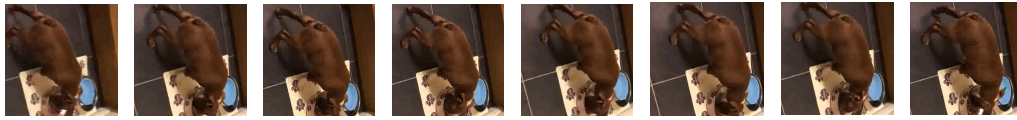

**Is there a canis in this video?**
(A)yes
(B)no
(C)not sure

Right answer: (A) yes

Chat-UniVi: (C) not sure.
mPLUG-Owl: (A) yes. Yes, there is a canis in the video. The brown dog is seen eating its dog food out of a bowl.
Valley: (A) Yes, there is a canis in the video.
VideoChat: (A) Yes, there is a dog in the video.
VideoChat2: (A) yes
Video-ChatGPT: (A) Yes there is a canis in the video.
Video-LLaMA: (A) yes.
Video-LLaVA: (A)

**Action Recognition**

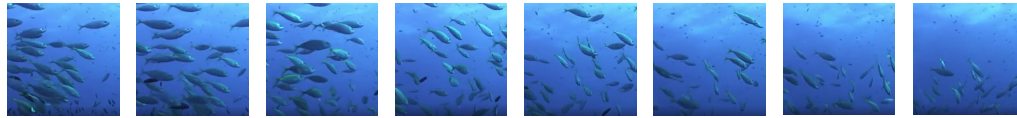

**What activity does the video depict?**
(A)Flying
(B)Spitting Venom
(C)Undergoing Chrysalis
(D)Swimming

Right answer: (D)Swimming

Chat-UniVi: (D) Swimming. The video shows a group of small fish swimming together in a school within the open sea, emphasizing the communal aspect of their behavior.
mPLUG-Owl: (A) Flying
Valley: (A) Flying.
VideoChat: (A) Flying
VideoChat2: (D) Swimming
Video-ChatGPT: (D) Swimming, as the video focuses on a fish swimming in water.
Video-LLaMA: (D) Swimming
Video-LLaVA: (C)

**Action Prediction**

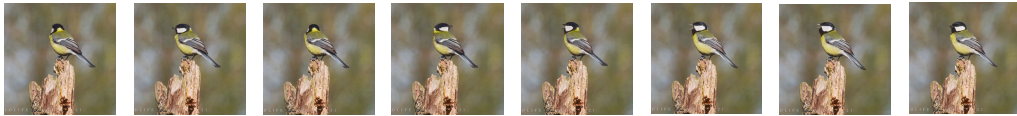

**What will the animal do next after the animal was chirping?**
(A)Hanging
(B)Drinking
(C)Surfacing
(D)Keeping still

Right answer: (D)Keeping still

Chat-UniVi: (C) Surfacing.
mPLUG-Owl: (A) Hanging.
Valley: (C) Surfacing, as this is the most appropriate response based on the given visual cues in the video. The squirrel is indeed surfing on its back.
VideoChat: (C) Surfacing
VideoChat2: (A) Hanging
Video-ChatGPT: (D) Keeping still, the bird is seen keeping still in the tree, possibly due to the camera's steady and long recording of the bird.
Video-LLaMA: (D) Keeping still
Video-LLaVA: (D)

Figure 12: The illustrative sample of the generated responses from different multimodal video models for "object recognition", "object existence", "action recogntion" and "action prediction" tasks. Green indicates correct answers, red indicates incorrect answers, and orange indicates that although the chosen option is correct, there are some errors in the response.

**Action Localization**

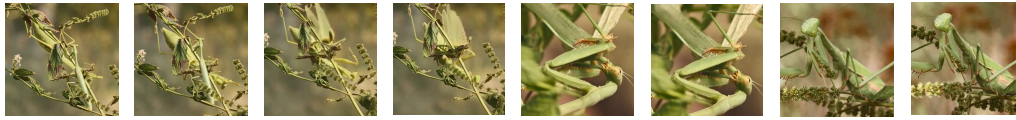

**Can you identify when the action 'keeping still' happens in the video?**
(A)At the beginning of the video.
(B)In the middle of the video.
(C)At the end of the video.
(D)Throughout the entire video.

**Right answer: (C)At the end of the video.**

Chat-UniVi: (B) In the middle of the video.
mPLUG-Owl: (C) At the end of the video, as the mantis is still after the plant it is on starts moving.
Valley: (D) Throughout the entire video.
VideoChat: (D) Throughout the entire video.
VideoChat2: (B) In the middle of the video.
Video-ChatGPT: (C) At the end of the video.
Video-LLaMA: (B) In the middle of the video.
Video-LLaVA: (A)

**Reasoning**

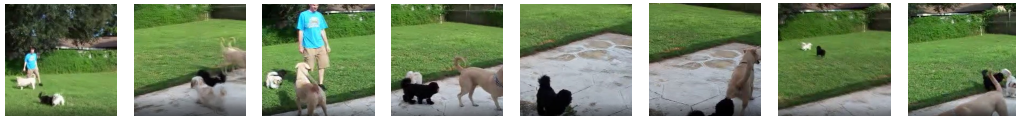

**why did the black dog ran after the white dog ran towards the grass near the end of the video?**
(A)it was chased
(B)brown dog took its toy
(C)play with ball
(D)playing with each other
(E)attracted by it

**Right answer: (D)playing with each other**

Chat-UniVi: (C) play with ball.
mPLUG-Owl: (D) Playing with each other. Both dogs are seen playing with a ball, which could have been the reason the white dog ran towards the grass and the black dog ran after it. Dogs love to play, and this type of playful interaction is common among them.
Valley: (A) the black dog was chased by the white dog running towards the grass.
VideoChat: (B) brown dog took its toy.
VideoChat2: (A) it was chased
Video-ChatGPT: (B) The black dog chased after the white dog who had taken its toy and was running towards the grass.
Video-LLaMA: (D) playing with each other
Video-LLaVA: (B)

**Object Count**

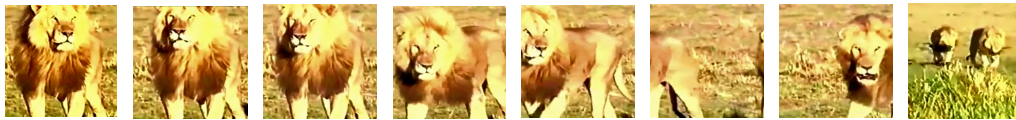

**how many lions are walking in the forest?**
(A)Three
(B)Five
(C)Four
(D)Two

**Right answer: (D)Two**

Chat-UniVi: (C) four.
mPLUG-Owl: (A) three\n\nThe video shows three lions walking in the forest.
Valley: (C) four
VideoChat: (A) three
VideoChat2: (D) two
Video-ChatGPT: (B) five. In the video, five lions are walking in different directions while the camera focuses on their actions. Although observing multiple lions walking could still be indicative of a pride, it is not possible to determine the exact number from a single image.
Video-LLaMA: (C) four lions are walking in the forest.
Video-LLaVA: (A)

**Social Interaction Analysis**

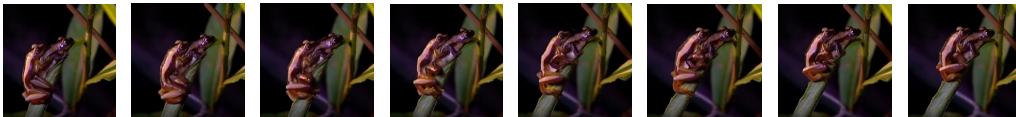

**Is the animal in this video hugging?**
(A)no
(B)yes
(C)not sure

**Right answer: (B)yes**

Chat-UniVi: (B) Yes, the animal in this video is hugging.
mPLUG-Owl: (C) Not sure. The animal in the video is not hugging..
Valley: (C) not sure\nPlease watch the video again and make sure you have observed the animal clearly before providing your answer.
VideoChat: (B) yes
VideoChat2: (B) yes
Video-ChatGPT: (B) Yes - The video shows an animal hugging an object while moving across the screen. Since the object is green, it is possible that it is a stalk with foliage or a plant.
Video-LLaMA: (B) yes.
Video-LLaVA: (A)

Figure 13: The illustrative sample of the generated responses from different multimodal video models for "action localization", "reasoning", "object count" and "social interation analysis" tasks. Green indicates correct answers, red indicates incorrect answers, and orange indicates that although the chosen option is correct, there are some errors in the response.

